# HIGH DENSITY ASSOCIATIVE MEMORIES[1]

Amir Dembo
Information Systems Laboratory, Stanford University
Stanford, CA 94305

Ofer Zeitouni
Laboratory for Information and Decision Systems
MIT, Cambridge, MA 02139

## ABSTRACT

A class of high density associative memories is constructed, starting from a description of desired properties those should exhibit. These properties include high capacity, controllable basins of attraction and fast speed of convergence. Fortunately enough, the resulting memory is implementable by an artificial Neural Net.

## INTRODUCTION

Most of the work on associative memories has been structure oriented; i.e., given a Neural architecture, efforts were directed towards the analysis of the resulting network. Issues like capacity, basins of attractions, etc. were the main objects to be analyzed cf., e.g. [1], [2], [3], [4] and references there, among others.

In this paper, we take a different approach; we start by explicitly stating the desired properties of the network, in terms of capacity, etc. Those requirements are given in terms of axioms (c.f. below). Then, we bring a synthesis method which enables one to design an architecture which will yield the desired performance. Surprisingly enough, it turns out that one gets rather easily the following properties:

(a) High capacity (unlimited in the continuous state-space case, bounded only by sphere-packing bounds in the discrete state case).

(b) Guaranteed basins of attractions in terms of the natural metric of the state space.

(c) High speed of convergence in the guaranteed basins of attraction.

Moreover, it turns out that the architecture suggested below is the only one which satisfies all our axioms ("desired properties")!

Our approach is based on defining a potential and following a descent algorithm (e.g., a gradient algorithm). The main design task is to construct such a potential (and, to a lesser extent, an implementation of the descent algorithm via a Neural network). In doing so, it turns out that, for reasons described below, it is useful to regard each desired memory location as a "particle" in the state space. It is natural to require now the following requirement from a

memory:

(P1) The potential should be linear w.r.t. adding particles in the sense that the potential of two particles should be the sum of the potentials induced by the individual particles (i.e., we do not allow interparticles interaction).

(P2) Particle locations are the only possible sites of stable memory locations.

(P3) The system should be invariant to translations and rotations of the coordinates.

We note that the last requirement is made only for the sake of simplicity. It is not essential and may be dropped without affecting the results.

In the sequel, we construct a potential which satisfies the above requirements. We refer the reader to [5] for details of the proofs, etc.

Acknowledgements. We would like to thank Prof. L.N. Cooper and C.M. Bachmann for many fruitful discussions. In particular, section 2 is part of a joint work with them ([6]).

## 2. HIGH DENSITY STORAGE MODEL

In what follows we present a particular case of a method for the construction of a high storage density neural memory. We define a function with an arbitrary number of minima that lie at preassigned points and define an appropriate relaxation procedure. The general case in presented in [5].

Let $\bar{x}_1, \ldots, \bar{x}_m$ be a set of m arbitrary distinct memories in $R^N$. The "energy" function we will use is:

$$\xi = -\frac{1}{L} \sum_{i=1}^{m} Q_i |\bar{\mu} - \bar{x}_i|^{-L} \tag{1}$$

where we assume throughout that $N \geq 3$, $L \geq (N-2)$, and $Q_i > 0$ and use $|\ldots|$ to denote the Euclidean distance. Note that for $L = 1$, $N=3$, $\xi$ is the electrostatic potential induced by negative fixed particles with charges $-Q_i$. This "energy" function possesses global minima at $\bar{x}_1, \ldots, \bar{x}_m$ (where $\xi(\bar{x}_i) = -\infty$) and has no local minima except at these points. A rigorous proof is presented in [5] together with the complete characterization of functions having this property.

As a relaxation procedure, we can choose any dynamical system for which $\xi$ is strictly decreasing, uniformly in compacts. In this instance, the theory of dynamical systems guarantees that for almost any initial data, the trajectory of the system converges to one of the desired points $\bar{x}^1, \ldots, \bar{x}^m$. However, to give concrete results and to further exploit the resemblance to electrostatic, consider the relaxation:

$$\dot{\bar{\mu}} = \bar{E}_{\bar{\mu}} = - \sum_{i=1}^{m} Q_i |\bar{\mu} - \bar{x}_i|^{-(L+2)} (\bar{\mu} - \bar{x}_i) \qquad (2)$$

where for $N=3$, $L=1$, equation (2) describes the motion of a positive test particle in the electrostatic field $\bar{E}_{\bar{\mu}}$ generated by the negative fixed charges $-Q_1, \ldots, -Q_m$ at $\bar{x}_1, \ldots, \bar{x}_m$.

Since the field $\bar{E}_{\bar{\mu}}$ is just minus the gradient of $\xi$, it is clear that along trajectories of (2), $d\xi/dt \leq 0$, with equality only at the fixed points of (2), which are exactly the stationary points of $\xi$. Therefore, using (2) as the relaxation procedure, we can conclude that entering at any $\bar{\mu}(0)$, the system converges to a stationary point of $\xi$. The space of inputs is partitioned into m domains of attraction, each one corresponding to a different memory, and the boundaries (a set of measure zero), on which $\bar{\mu}(0)$ will converge to a saddle point of $\xi$.

We can now explain why $\xi_{\bar{\mu}}$ has no spurious local minima, at least for $L=1$, $N=3$, using elementary physical arguments. Suppose $\xi$ has a spurious local minima at $\bar{y} \neq \bar{x}_1, \ldots, \bar{x}_m$, then in a small neighborhood of $\bar{y}$ which does not include any of the $\bar{x}_i$, the field $\bar{E}_{\bar{\mu}}$ points towards $\bar{y}$. Thus, on any closed surface in that neighborhood, the integral of the normal inward component of $\bar{E}_{\bar{\mu}}$ is positive. However, this integral is just the total charge included inside the surface, which is zero. Thus we arrive at a contradiction, so $\bar{y}$ can not be a local minimum.

We now have a relaxation procedure, such that almost any $\bar{\mu}(0)$ is attracted by one of the $\bar{x}_i$, but we have not yet specified the shapes of the basins of attraction. By varying the charges $Q_i$, we can enlarge one basin of attraction at the expense of the others (and vice versa).

Even when all of the $Q_i$ are equal, the position of the $\bar{x}_i$ might cause $\bar{\mu}(0)$ not to converge to the closest memory, as emphasized in the example in fig. 1. However, let $r = \min_{1 \leq i \neq j \leq m} |\bar{x}_i - \bar{x}_j|$ be the minimal distance between any two memories; then if $|\bar{\mu}(0) - \bar{x}_i| \leq \frac{r}{(1 + 3^{1/k})}$ it can be shown that $\bar{\mu}(0)$ will converge to $\bar{x}_i$, (provided that $k = \frac{L+1}{N+1}$ $\geq 1$). Thus, if the memories are densely packed in a hypersphere, by choosing k large enough (i.e. enlarging the parameter L), convergence to the closest memory for any "interesting" input, that is an input $\bar{\mu}(0)$ with a distinct closest memory, is guaranteed. The detailed proof of the above property is given in [5]. It is based on bounding the number of $\bar{x}_j$, $j \neq i$, in a hypersphere of radius $R(R \geq r)$ around $\bar{x}_i$, by $[2R/r + 1]^N$, then bounding the magnitude of the field induced by any $\bar{x}_j$, $j \neq i$, on the boundary of such a hypersphere by $(R - |\bar{\mu}(0) - \bar{x}_i|)^{-(L+1)}$, and finally integrating to show that for $|\bar{\mu}(0) - \bar{x}_i| \leq \frac{\theta r}{(1 + 3^{1/k})}$, with $\theta < 1$, the convergence of $\bar{\mu}(0)$ to $\bar{x}_i$ is within finite time T, which behaves like $\theta^{L+2}$ for $L \gg 1$ and $\theta < 1$ and fixed. Intuitively the reason for this behaviour is the short-range nature of the fields used in equation (2). Because of this, we also expect extremely low convergence rate for inputs $\bar{\mu}(0)$ far away from all of the $\bar{x}_i$.

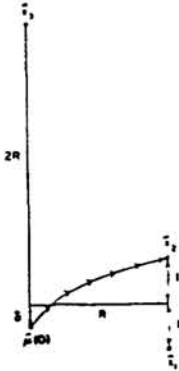

Figure 1

$R \gg 1$ and $\delta \ll 1$

The radial nature of these fields suggests a way to overcome this difficulty, that is to increase the convergence rate from points very far away, without disturbing all of the aforementioned desirable properties of the model. Assume that we know in advance that all of the $\bar{x}_i$ lie inside some large hypersphere S around the origin. Then, at any point $\bar{\mu}$ outside S, the field $\bar{E}_{\bar{\mu}}$ has a positive projection radially into S. By adding a long-range force to $\bar{E}_{\bar{\mu}}$, effective only outside of S, we can hasten the movement towards S, from points far away, without creating additional minima inside of S. As an example the force $(-\bar{\mu}$ for $\bar{\mu} \notin S;$ 0 for $\bar{\mu}\ \varepsilon\ S)$ will pull any test input $\bar{\mu}(0)$ to the boundary of S within the small finite time $T \approx 1/|S|$, and from then on the system will behave inside S according to the original field $\bar{E}_{\bar{\mu}}$.

Up to this point, our derivations have been for a continuous system, but from it we can deduce a discrete system. We shall do this mainly for a clearer comparison between our high density memory model and the discrete version of Hopfield's model. Before continuing in that direction, note that our continuous system has unlimited storage capacity unlike Hopfield's continuous system, which like his discrete model, has limited capacity.

For the discrete system, assume that the $\bar{x}_i$ are composed of elements $\pm 1$ and replace the Euclidean distance in (1) with the normalized Hamming distance $|\bar{\mu}_1 - \bar{\mu}_2| = \frac{1}{N} \sum_{j=1}^{N} |\mu_j^1 - \mu_j^2|$. This places the vectors $\bar{x}_i$ on the unit hypersphere.

The relaxation process for the discrete system will be of the type defined in Hopfield's model in [1] . Choose at random a component to be updated (that is, a neighbor $\bar{\mu}'$ of $\bar{\mu}$ such that $|\bar{\mu}' - \bar{\mu}| = 2/N$), calculate the "energy" difference, $\delta\xi = \xi(\bar{\mu}')-\xi(\bar{\mu})$, and only if $\delta\xi < 0$, change this component, that is:

$$\mu_i \to \mu_i \ \text{sign}(\xi(\bar{\mu}') - \xi(\bar{\mu})), \qquad (3)$$

where $\xi(\bar{\mu})$ is the potential energy in (1). Since there is a finite number of possible $\bar{\mu}$ vectors $(2^N)$, convergence in finite time is guaranteed.

This relaxation procedure is rigid since the movement is limited to points with components $\pm 1$. Therefore, although the local minima of $\xi(\bar{\mu})$ defined in (2) are only at the desired points $\bar{x}_i$, the relaxation may get stuck at some $\bar{\mu}$ which is not a stationary point of $\xi(\bar{\mu})$. However, the short range behaviour of the potential $\xi(\bar{\mu})$, unlike the long-range behavior of the quadratic potential used by Hopfield, gives

rise to results similar to those we have quoted for the continuous model (equation (1)).

Specifically, let the stored memories $\bar{x}_1,\ldots,\bar{x}_m$ be separated from one another by having at least $\rho N$ different components ($0 < \rho \le 1/2$ and $\rho$ fixed), and let $\bar{\mu}(0)$ agree up to at least one $\bar{x}_i$ with at most $\theta\rho N$ errors between them ($0 \le \theta < 1/2$, with $\theta$ fixed), then $\bar{\mu}(0)$ converges monotonically to $\bar{x}_i$ by the relaxation procedure given in equation (3).

This result holds independently of m, provided that N is large

enough (typically, $N\rho \ln(\frac{1-\theta}{\theta}) \ge 1$) and L is chosen so that $\frac{N}{L} \le \ln(\frac{1-\theta}{\theta})$

The proof is constructed by bounding the cummulative effect of terms

$|\bar{\mu} - \bar{x}_j|^{-L}$, $j \ne i$, to the energy difference $\delta\xi$ and showing that it is dominated by $|\bar{\mu} - \bar{x}_i|^{-L}$. For details, we refer the reader again to [5].

Note the importance of this property: unlike the Hopfield model which is limited to $m \le N$, the suggested system is optimal in the sense of Information Theory, since for every set of memories $\bar{x}_1,\ldots,\bar{x}_m$ separated from each other by a Hamming distance $\rho N$, up to $1/2 \, \rho N$ errors in the input can be corrected, provided that N is large and L properly chosen.

As for the complexity of the system, we note that the nonlinear operation $a^{-L}$, for $a > 0$ and L integer (which is at the heart of our system computationally) is equivalent to $e^{-L\ln(a)}$ and can be implemented, therefore, by a simple electrical circuit composed of diodes, which have exponential input-output characteristics, and resistors, which can carry out the necessary multiplications (cf. the implementation of section 3).

Further, since both $|\bar{x}_i|$ and $|\bar{\mu}|$ are held fixed in the discrete system, where all states are on the unit hypersphere, $|\bar{\mu} - \bar{x}_i|^2$ is equivalent to the inner product of $\bar{\mu}$ and $\bar{x}_i$, up to a constant.

To conclude, the suggested model involves about $m \cdot N$ multiplications, followed by m nonlinear operations, and then $m \cdot N$ additions. The original model of Hopfield involves $N^2$ multiplications and additions, and then N nonlinear operations, but is limited to $m \le N$. Therefore, whenever the Hopfield model is applicable the complexity of both models is comparable.

## 3. IMPLEMENTATION

We propose below one possible network which implements the discrete time and space version of the model described above. An implementation for the ocntinuous time case, which is even simpler, is also hinted. We point out that the implementation described below is by no means unique, (and maybe even not the simplest one). Moreover, the "neurons" used are artificial neurons which perform various tasks, as follows: There are (N+1) neurons which are delay elements, and m pointwise non-linear functions (which may be interpreted as delay-less, intermediate neurons). There are $mN$ synaptic connections between those two layers of neurons. In addition, as in the Hopfield model, we have at each iteration to specify (either deterministically or stochastically) which coordinate are we updating. To do that, we use an N dimensional "control register" whose content is always a unit vector of $\{0, 1\}^N$ (and the location of the '1' will denote the next coordiante to be changed). This vector may be varied from instant n to n + 1 either by shift ("sequential coordinate update") or at random.

Let $\Delta_i$, $i \leq i \leq N$ be the i-th output of the "control" register, $x_i$, $1 \leq i \leq N$ and V be the (N+1) neurons inputs and $\tilde{x}_i = x_i(1-2\Delta_i)$ the corresponding outputs (where $\tilde{x}_i$, $x_i \varepsilon \{+1,-1\}$, $\Delta_i \varepsilon \{0,1\}$, but V is a real number), $\phi_j$, $1 \leq j \leq m$ be the input of the j-th intermediate neuron ($-1 \leq \phi_j \leq 1$), $\eta_j = -(1-\phi_j)^{-L}$ be its output, and $W_{ji} = U_i^{(j)}/N$ be the synaptic weight of the ij - th synapsis, where $U_i^{(j)}$ refers here to the i-th element of the j-th memory.

The system's equations are:

$$\tilde{x}_i = x_i(1 - 2\Delta_i) \qquad 1 \leq i \leq N \qquad (4a)$$

$$\phi_j = \sum_{i=1}^{N} W_{ji}\tilde{x}_i \qquad 1 \leq j \leq m \qquad (4b)$$

$$\eta_j = -(1 - \phi_j)^{-L} \qquad 1 \leq j \leq m \qquad (4c)$$

$$\tilde{V} = \sum_{j=1}^{m} \eta_j \qquad (4d)$$

$$S = \frac{1}{2}(1-\text{sign}(\tilde{V} - V)) \qquad (4e)$$

$$x_i \leftarrow x_i + S\tilde{x}_i \qquad 1 \leq i \leq N \qquad (4f)$$

$$V \leftarrow V + S\tilde{V} \qquad (4g)$$

The system is initialized by $x_i = x_i(0)$ (the probe vector), and $V = +\infty$. A block diagram of this sytem appears in Fig. 2. Note that we made use of N + m + 1 neurons and 0(Nm) connections.

As for the continuous time case (with memories on the unit sphere) we will get the equations:

$$\dot{x}_i + 2m\,\tilde{V}x_i = LN\sum_{j=1}^{m} W_{ji}\eta_j, \qquad\qquad 1 \leq i \leq N \quad (5a)$$

$$\phi_j = N\sum_{i=1}^{N} W_{ji}x_i, \quad \delta = \sum_{i=1}^{N} x_i^2, \qquad 1 \leq j \leq m \quad (5b)$$

$$\eta_j = (1 + \delta - 2\phi_j)^{-(\frac{L}{2}+1)}, \qquad\qquad 1 \leq j \leq m \quad (5c)$$

$$\tilde{V} = \sum_{j=1}^{m} \eta_j \qquad\qquad\qquad\qquad (5d)$$

with similar interpretation (here there is no 'control' register as all components are updated continuously).

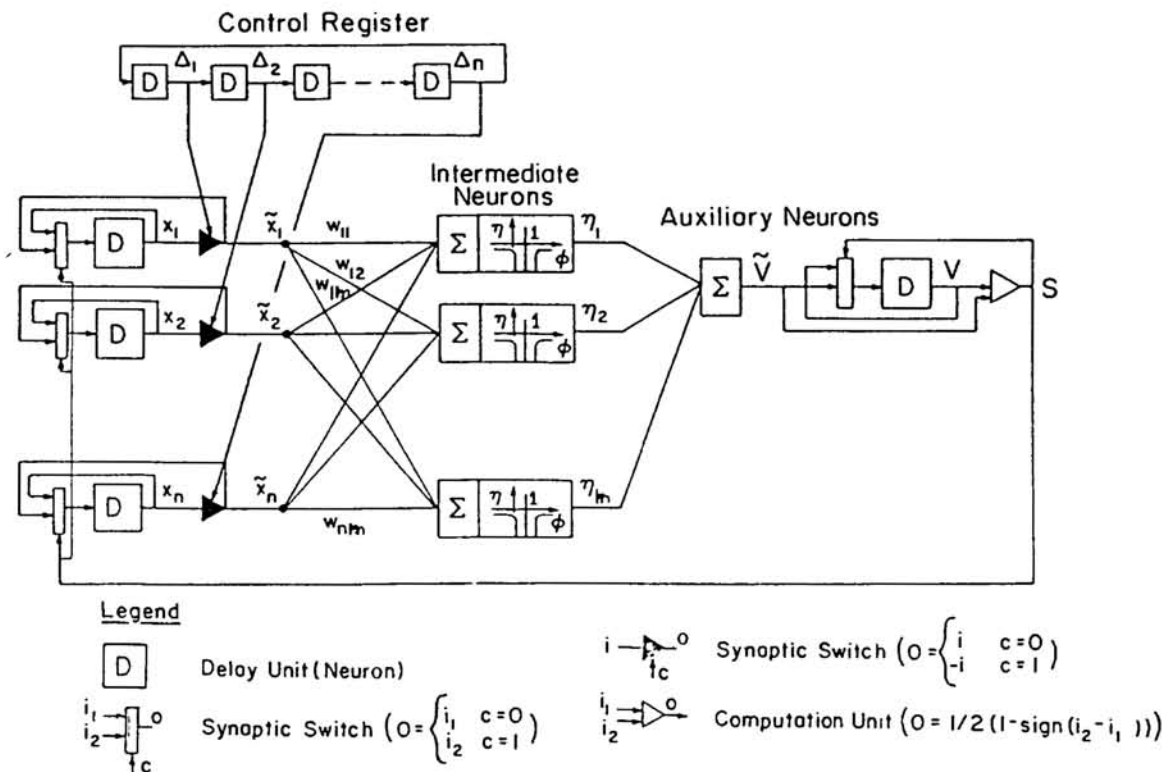

Figure 2 Neural Network Implementation

## Footnotes

[1]An expanded version of this work has been submitted to Phys. Rev. A. This work was carried out at the Center for Neural Science, Brown University.

## REFERENCES

1. J.J. Hopfield, "Neural Networks and Physical Systems with Emergent Collective Computational Abilities", Proc. Nat. Acad. Sci. U.S.A., Vol. 79 (1982), pp. 2554-2558.

2. R.J. McEliece, et al., "The Capacity of the Hopfield Associative Memory", IEEE Trans. on Inf. Theory, Vol. IT-33 (1987), pp. 461-482.

3. A. Dembo, "On the Capacity of the Hopfield Memory", submitted, IEEE Trans. on Inf. Theory.

4. Kohonen, T., Self Organization and Associative Memory, Springer, Berlin, 1984.

5. Dembo, A. and Zeitouni, O., General Potential Surfaces and Neural Networks, submitted, Phys. Rev. A.

6. Bachmann, C.M., Cooper, L.N., Dembo, A. and Zeitouni, O., A relazation Model for Memory with high storage density, to appear, Proc. Natl. Ac. Science.
